# Bayesian Synchronous Grammar Induction

**Phil Blunsom, Trevor Cohn, Miles Osborne**
School of Informatics, University of Edinburgh
10 Crichton Street, Edinburgh, EH8 9AB, UK
{pblunsom,tcohn,miles}@inf.ed.ac.uk

## Abstract

We present a novel method for inducing synchronous context free grammars (SCFGs) from a corpus of parallel string pairs. SCFGs can model equivalence between strings in terms of substitutions, insertions and deletions, and the reordering of sub-strings. We develop a non-parametric Bayesian model and apply it to a machine translation task, using priors to replace the various heuristics commonly used in this field. Using a variational Bayes training procedure, we learn the latent structure of translation equivalence through the induction of synchronous grammar categories for phrasal translations, showing improvements in translation performance over maximum likelihood models.

## 1  Introduction

A recent trend in statistical machine translation (SMT) has been the use of synchronous grammar based formalisms, permitting polynomial algorithms for exploring exponential forests of translation options. Current state-of-the-art synchronous grammar translation systems rely upon heuristic relative frequency parameter estimates borrowed from phrase-based machine translation[1, 2]. In this work we draw upon recent Bayesian models of monolingual parsing [3, 4] to develop a generative synchronous grammar model of translation using a hierarchical Dirichlet process (HDP) [5].

There are two main contributions of this work. The first is that we include sparse priors over the model parameters, encoding the intuition that source phrases will have few translations, and also addressing the problem of overfitting when using long multi-word translations pairs. Previous models have relied upon heuristics to implicitly bias models towards such distributions [6]. In addition, we investigate different priors based on standard machine translation models. This allows the performance benefits of these models to be combined with a principled estimation procedure.

Our second contribution is the induction of categories for the synchronous grammar using a HDP prior. Such categories allow the model to learn the latent structure of translational equivalence between strings, such as a preference to reorder adjectives and nouns when translating between French to English or to encode that a phrase pair should be used at the beginning or end of a sentence. Automatically induced non-terminal symbols give synchronous grammar models increased power over single non-terminal systems such as [2], while avoiding the problems of relying on noisy domain-specific parsers, as in [7]. As the model is non-parametric, the HDP prior will provide a bias towards parameter distributions using as many, or as few, non-terminals as necessary to model the training data. Following [3] we optimise a truncated variational bound on the true posterior distribution.

We evaluate the model on both synthetic data, and the real task of translating from Chinese to English, showing improvements over a maximum likelihood estimate (MLE) model. We focus on modelling the generation of a translation for a source sentence, putting aside for further work integration with common components of a state-of-the-art translation system, such as a language model and minimum error rate training [6].

While we are not aware of any previous attempts to directly induce synchronous grammars with more than a single category, a number of generatively trained machine translation models have been

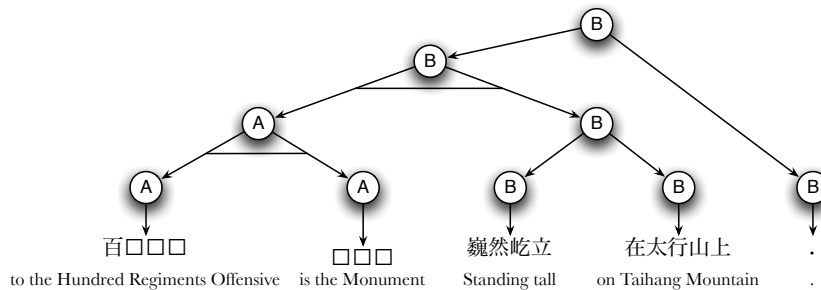

百□□□ □□□ 巍然屹立 在太行山上 .
to the Hundred Regiments Offensive  is the Monument  Standing tall  on Taihang Mountain  .

Figure 1: An example SCFG derivation from a Chinese source sentence which yields the English sentence: *"Standing tall on Taihang Mountain is the Monument to the Hundred Regiment Offensive."* (Cross-bars indicate that the child nodes have been reordered in the English target.)

proposed. [8] described the ITG subclass of SCFGs and performed many experiments using MLE training to induce translation models on small corpora. Most subsequent work with ITG grammars has focused on the sub-task of word alignment [9], rather than actual translation, and has continued to use MLE trained models. A notable recent exception is [10] who used Dirichlet priors to smooth an ITG alignment model. Our results clearly indicate that MLE models considerably overfit when used to estimate synchronous grammars, while the judicious use of priors can alleviate this problem. This result raises the prospect that many MLE trained models of translation (e.g. [7, 11, 12]), previously dismissed for under-performing heuristic approaches, should be revisited.

## 2 Synchronous context free grammar

A synchronous context free grammar (SCFG, [13]) describes the generation of pairs of strings. A string pair is generated by applying a series of paired context-free rewrite rules of the form, $X \rightarrow \langle \gamma, \phi \rangle$, where $X$ is a non-terminal, $\gamma$ and $\phi$ are strings of terminals and non-terminals and $\sim$ specifies a one-to-one alignment between non-terminals in $\gamma$ and $\phi$. In the context of SMT, by assigning the source and target languages to the respective sides of a SCFG it is possible to describe translation as the process of parsing the source sentence, while generating the target translation [2].

In this paper we only consider binary normal-form SCFGs which allow productions to rewrite as either a pair of a pair of non-terminals, or a pair of non-empty terminal strings (these may span multiple words). Such grammars are equivalent to the inversion transduction grammars presented in [8]. Note however that our approach is general and could be used with other synchronous grammar transducers (e.g., [7]). The binary non-terminal productions can specify that the order of the child non-terminals is the same in both languages (a *monotone* production), or is reversed (a *reordering* production). Monotone and reordering rules are written:

$$Z \rightarrow X_{\boxed{1}} Y_{\boxed{2}} \mid X_{\boxed{1}} Y_{\boxed{2}} \quad \text{and} \quad Z \rightarrow X_{\boxed{1}} Y_{\boxed{2}} \mid Y_{\boxed{2}} X_{\boxed{1}}$$

respectively, where $X$, $Y$ and $Z$ are non-terminals and the boxed indices denote the alignment. Without loss of generality, here we add the restriction that non-terminals on the source and target sides of the grammar must have the same category. Although conceptually simple, a binary normal-form SCFGs can still represent a wide range of linguistic phenomena required for translation [8].

Figure 1 shows an example derivation for Chinese to English. The grammar in this example has non-terminals $A$ and $B$ which distinguish between translation phrases which permit re-orderings.

## 3 Generative Model

A sequence of SCFG rule applications which produces both a source and a target sentence is referred to as a *derivation*, denoted $\mathbf{z}$. The generative process of a derivation in our model is described in Table 1. First a start symbol, $z_1$, is drawn, followed by its rule type. This rule type determines if the symbol will rewrite as a source-target translation pair, or a pair of non-terminals with either monotone or reversed order. The process then recurses to rewrite each pair of child non-terminals.

<table_top>

| HDP-SCFG | |
|---|---|
</table_top>

**HDP-SCFG**

$\pi \| \alpha \sim \mathrm{GEM}(\alpha)$     (Draw top-level constituent prior distribution)

$\phi^S \| \alpha^S, \pi \sim \mathrm{DP}(\alpha^S, \pi)$     (Draw start-symbol distribution)

$\phi^T_z \| \alpha^Y \sim \mathrm{Dirichlet}(\alpha^Y)$     (Draw rule-type parameters)

$\phi^M_z \| \alpha^M, \pi \sim \mathrm{DP}(\alpha^M, \pi\pi^T)$     (Draw monotone binary production parameters)

$\phi^R_z \| \alpha^R, \pi \sim \mathrm{DP}(\alpha^R, \pi\pi^T)$     (Draw reordering binary production parameters)

$\phi^E_z \| \alpha^E, P_0 \sim \mathrm{DP}(\alpha^E, P_0)$     (Draw emission production parameters)

$z_1 \| \phi^S \sim \mathrm{Multinomial}(\phi^S)$     (First draw the start symbol)

For each node $i$ in the synchronous derivation **z** with category $z_i$:

   $t_i \| \phi^T_{z_i} \sim \mathrm{Multinomial}(\phi^T_{z_i})$     (Draw a rule type)

   if $t_i = \mathrm{Emission}$ then:

      $\langle \mathbf{e}, \mathbf{f} \rangle \| \phi^E_{z_i} \sim \mathrm{Multinomial}(\phi^E_{z_i})$     (Draw source and target phrases)

   if $t_i = \mathrm{Monotone\ Production}$ then:

      $\langle z_{l\boxed{1}} z_{r\boxed{2}}, z_{l\boxed{1}} z_{r\boxed{2}} \rangle \| \phi^M_{z_i} \sim \mathrm{Multinomial}(\phi^M_{z_i})$     (Draw left and right (source) child constituents)

   if $t_i = \mathrm{Reordering\ Production}$ then:

      $\langle z_{l\boxed{1}} z_{r\boxed{2}}, z_{r\boxed{2}} z_{l\boxed{1}} \rangle \| \phi^R_{z_i} \sim \mathrm{Multinomial}(\phi^R_{z_i})$     (Draw left and right (source) child constituents)

Table 1: Hierarchical Dirichlet process model of the production of a synchronous tree from a SCFG.

This continues until no non-terminals are remaining, at which point the derivation is complete and the source and target sentences can be read off. When expanding a production each decision is drawn from a multinomial distribution specific to the non-terminal, $z_i$. This allows different non-terminals to rewrite in different ways – as an emission, reordering or monotone production. The prior distribution for each binary production is parametrised by $\pi$, the top-level stick-breaking weights, thereby ensuring that each production draws its children from a shared inventory of category labels.

The parameters for each multinomial distributions are themselves drawn from their corresponding prior. The hyperparameters, $\alpha, \alpha^S, \alpha^Y, \alpha^M, \alpha^R$, and $\alpha^E$, encode prior knowledge about the sparsity of each distribution. For instance, we can encode a preference towards longer or short derivations using $\alpha^Y$, and a preference for sparse or dense translation lexicons with $\alpha^E$. To simplify matters we assume a single hyperparameter for productions, i.e. $\alpha^P \triangleq \alpha^S = \alpha^M = \alpha^R$. In addition to allowing for the incorporation of prior knowledge about sparsity, the priors have been chosen to be conjugate to the multinomial distribution. In the following sections we describe and motivate our choices for each one of these distributions.

## 3.1 Rule type distribution

The rule type distribution determines the relative likelihood of generating a terminal string pair, a monotone production, or a reordering. Synchronous grammars that allow multiple words to be emitted at the leaves of a derivation are prone to focusing probability mass on only the longest translation pairs, i.e. if a training set sentence pair can be explained by many short translation pairs, or a few long ones the maximum likelihood solution will be to use the longest pairs. This issue is manifested by the rule type distribution assigning a high probability to emissions versus either of the binary productions, resulting in short flat derivations with few productions. We can counter this tendency by assuming a prior distribution that allows us to temper the model's preference for short derivations with large translation pairs. We do so by setting the concentration parameter, $\alpha^Y$, to a number greater than one which smooths the rule type distribution.

## 3.2 Emission distribution

The Dirichlet process prior on the terminal emission distribution serves two purposes. Firstly the prior allows us to encode the intuition that our model should have few translation pairs. The translation pairs in our system are induced from noisy data and thus many of them will be of little use. Therefore a sparse prior should lead to these noisy translation pairs being assigned probabilities

close to zero. Secondly, the base distribution $P_0$ of the Dirichlet process can be used to include sophisticated prior distributions over translation pairs from other popular models of translation. The two structured priors we investigate in this work are IBM model 1, and the relative frequency count estimators from phrase based translation:

**IBM Model 1** ($P_0^{m1}$)   IBM Model 1 [14] is a word based generative translation model that assigns a joint probability to a source and target translation pair. The model is based on a noisy channel in which we decompose the probability of $\mathbf{f}$ given $\mathbf{e}$ from the language model probability of $\mathbf{e}$. The conditional model assumes a latent alignment from words in $\mathbf{e}$ to those in $\mathbf{f}$ and that the probability of word-to-word translations are independent:

$$P_0^{m1}(\mathbf{f}, \mathbf{e}) = P^{m1}(\mathbf{f}|\mathbf{e}) \times P(\mathbf{e}) = P(\mathbf{e}) \times \frac{1}{(|\mathbf{e}| + 1)^{|\mathbf{f}|}} \times \prod_{j=1}^{|\mathbf{f}|} \sum_{i=0}^{|\mathbf{e}|} p(f_j|e_i) \,,$$

where $e_0$ represents word insertions. We use a unigram language model for the probability $P(\mathbf{e})$, and train the parameters $p(f_j|e_i)$ using a variational approximation, similar to that which is described in Section 3.4.

Model 1 allows us to assign a prior probability to each translation pair in our model. This prior suggests that lexically similar translation pairs should have similar probabilities. For example, if the French-English pairs *(chapeau, cap)* and *(rouge, red)* both have high probability, then the pair *(chapeau rouge, red cap)* should also.

**Relative frequency** ($P_0^{RF}$)   Most statistical machine translation models currently in use estimate the probabilities for translation pairs using a simple relative frequency estimator. Under this model the joint probability of a translation pair is simply the number of times the source was observed to be aligned to the target in the word aligned corpus normalised by the total number of observed pairs:

$$P_0^{RF}(\mathbf{f}, \mathbf{e}) = \frac{C(\mathbf{f}, \mathbf{e})}{C(*, *)} \,,$$

where $C(*, *)$ is the total number of translation pair alignments observed. Although this estimator doesn't take into account any generative process for how the translation pairs were observed, and by extension of the arguments for tree substitution grammars is biased and inconsistent [15], it has proved effective in many state-of-the-art translation systems.[1]

### 3.3   Non-terminal distributions

We employ a structured prior for binary production rules inspired by similar approaches in monolingual grammar induction [3, 4]. The marginal distribution over non-terminals, $\pi$, is drawn from a stick-breaking prior [5]. This generates an infinite vector of scalars which sum to one and whose expected values decrease geometrically, with the rate of decay being controlled by $\alpha$. The parameters of the start symbol distribution are drawn from a Dirichlet process parametrised by the stick-breaking weights, $\pi$. In addition, both the monotone and reordering production parameters are drawn from a Dirichlet process parameterised by the matrix of the expectations for each pair of non-terminals, $\pi\pi^T$, assuming independence in the prior. This allows the model to prefer grammars with few non-terminal labels and where each non-terminal has a sparse distribution over productions.

### 3.4   Inference

Previous work with monolingual HDP-CFG grammars have employed either Gibbs sampling [4] or variational Bayes [3] approaches to inference. In this work we follow the mean-field approximation presented in [16, 3], truncating the top-level stick-breaking prior on the non-terminals and optimising a variational bound on the probability of the training sample. The mean-field approach offers better scaling and convergence properties than a Gibbs sampler, at the expense of increased approximation.

First we start with our objective, the likelihood of the observed string pairs, $\mathbf{x} = \{(\mathbf{e}, \mathbf{f})\}$:

$$\log p(\mathbf{x}) = \log \int d\theta \sum_{\mathbf{z}} p(\theta)p(\mathbf{x}, \mathbf{z}|\theta) \geq \int d\theta \sum_{\mathbf{z}} q(\theta, \mathbf{z}) \log \frac{p(\theta)p(\mathbf{x}, \mathbf{z}|\theta)}{q(\theta, \mathbf{z})} \,,$$

where $\theta = (\pi, \phi^S, \phi^M, \phi^R, \phi^E, \phi^T)$ are our model parameters and $\mathbf{z}$ are the hidden derivations. We bound the above using Jensen's inequality to move the logarithm (a convex function) inside the integral and sum, and introduce the mean-field distribution $q(\theta, \mathbf{z})$. Assuming this distribution factorises over the model parameters and latent variables, $q(\theta, \mathbf{z}) = q(\theta)q(\mathbf{z})$,

$$\log p(\mathbf{x}) \geq \int d\theta q(\theta) \left( \log \frac{p(\theta)}{q(\theta)} + \sum_{\mathbf{z}} q(\mathbf{z}) \log \frac{p(\mathbf{x}, \mathbf{z}|\theta)}{q(\mathbf{z})} \right) \triangleq \mathcal{F}(q(\theta), q(\mathbf{z})) \ .$$

Upon taking the functional partial derivatives of $\mathcal{F}(q(\theta), q(\mathbf{z}))$ and equating to zero, we obtain sub-normalised summary weights for each of the factorised variational distributions: $W_i \triangleq \exp\{\mathbb{E}_{q(\phi)}[\log \phi_i]\}$. For the monotone and reordering distributions these become:

$$W_z^M(z_l, z_r) = \frac{\exp\{\psi\left(C\left(z \to \langle z_{l\boxed{1}} z_{r\boxed{2}}, z_{l\boxed{1}} z_{r\boxed{2}}\rangle\right) + \alpha^P \pi_{z_l} \pi_{z_r}\right)\}}{\exp\{\psi\left(C\left(z \to \langle *_{\boxed{1}}*_{\boxed{2}}, *_{\boxed{1}}*_{\boxed{2}}\rangle\right) + \alpha^P\right)\}}$$

$$W_z^R(z_l, z_r) = \frac{\exp\{\psi\left(C\left(z \to \langle z_{l\boxed{1}} z_{r\boxed{2}}, z_{r\boxed{2}} z_{l\boxed{1}}\rangle\right) + \alpha^P \pi_{z_l} \pi_{z_r}\right)\}}{\exp\{\psi\left(C\left(z \to \langle *_{\boxed{1}}*_{\boxed{2}}, *_{\boxed{2}}*_{\boxed{1}}\rangle\right) + \alpha^P\right)\}} \ ,$$

where $C(z \to \cdots)$ is the expected count of rewriting symbol $z$ using the given production. The starred rewrites in the denominators indicate a sum over any monotone or reordering production, respectively. The weights for the rule-type and emission distributions are defined similarly. The variational training cycles between optimising the $q(\theta)$ distribution by re-estimating the weights $W$ and the stick-breaking prior $\pi$, then using these estimates, with the inside-outside dynamic programming algorithm, to calculate the $q(\mathbf{z})$ distribution. Optimising the top-level stick-breaking weights has no closed form solution as a dependency is induced between the GEM prior and production distributions. [3] advocate using a gradient projection method to locally optimise this function. As our truncation levels are small, we instead use Monte-Carlo sampling to estimate a global optimum.

### 3.5   Prediction

The predictive distribution under our Bayesian model is given by:

$$p(\mathbf{z}|\mathbf{x}, \mathbf{f}) = \int d\theta \, p(\theta|\mathbf{x})p(\mathbf{z}|\mathbf{f}, \theta) \approx \int d\theta \, q(\theta)p(\mathbf{z}|\mathbf{f}, \theta) \geq \exp \int d\theta \, q(\theta) \log p(\mathbf{z}|\mathbf{f}, \theta) \ ,$$

where $\mathbf{x}$ is the training set of parallel sentence pairs, $\mathbf{f}$ is a testing source sentence and $\mathbf{z}$ its derivation.[2] Calculating the predictive probability even under the variational approximation is intractable, therefore we bound the approximation following [16]. The bound can then be maximised to find the best derivation, $\mathbf{z}$, with the Viterbi algorithm, using the sub-normalised $W$ parameters from the last $E$ step of variational Bayes training as the model parameters.

## 4   Evaluation

We evaluate our HDP-SCFG model on both synthetic and real-world translation tasks.

**Recovering a synthetic grammar**   This experiment investigates the ability of our model to recover a simple synthetic grammar, using the minimum number of constituent categories. Ten thousand training pairs were generated from the following synthetic grammar, with uniform weights, which includes both reordering and ambiguous terminal distributions:

$$S \to \langle A_{\boxed{1}} A_{\boxed{2}}, \ A_{\boxed{1}} A_{\boxed{2}}\rangle \qquad\qquad A \to \langle a, a\rangle | \langle b, b\rangle | \langle c, c\rangle$$
$$S \to \langle B_{\boxed{1}} B_{\boxed{2}}, \ B_{\boxed{2}} B_{\boxed{1}}\rangle \qquad\qquad B \to \langle d, d\rangle | \langle e, e\rangle | \langle f, f\rangle$$
$$S \to \langle C_{\boxed{1}} C_{\boxed{2}}, \ C_{\boxed{1}} C_{\boxed{2}}\rangle \qquad\qquad C \to \langle g, g\rangle | \langle h, h\rangle | \langle i, i\rangle$$

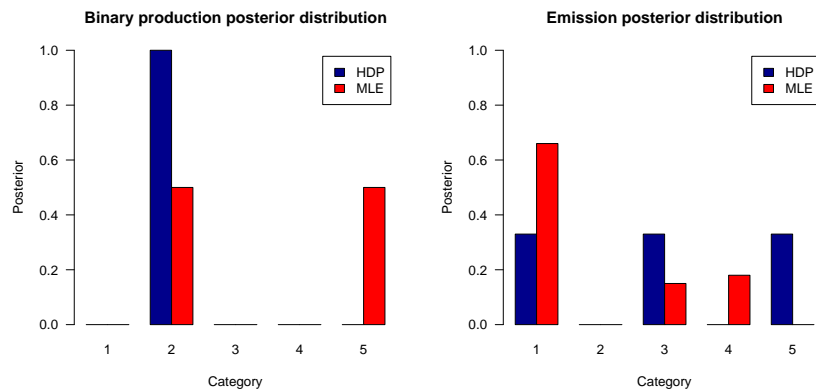

Figure 2: Synthetic grammar experiments. The HDP model correctly allocates a single binary production non-terminal and three equally weighted emission non-terminals.

|  | Training | | Development | | Test | |
|---|---|---|---|---|---|---|
| Sentences | Chinese | English | Chinese | English | Chinese | English |
| Sentences | 33164 | | 500 | | 506 | |
| Segments/Words | 253724 | 279104 | 3464 | 3752 | 3784 | 3823 |
| Av. Sentence Length | 7 | 8 | 6 | 7 | 7 | 7 |
| Longest Sentence | 41 | 45 | 58 | 62 | 61 | 56 |

Table 2: Chinese to English translation corpus statistics.

Figure 2 shows the emission and production distributions produced by the HDP-SCFG model,[3] as well as an EM trained maximum likelihood (MLE) model. The variational inference for the HDP model was truncated at five categories, likewise the MLE model was trained with five categories.

The hierarchical model finds the correct grammar. It allocates category $2$ to the $S$ category, giving it a $\frac{2}{3}$ probability of generating a monotone production $(A,C)$, versus $\frac{1}{3}$ for a reordering $(B)$. For the emission distribution the HDP model assigns category $1$ to $A$, $3$ to $B$ and $5$ to $C$, each of which has a posterior probability of $\frac{1}{3}$. The stick-breaking prior biases the model towards using a small set of categories, and therefore the model correctly uses only four categories, assigning zero posterior probability mass to category $4$.

The MLE model has no bias for small grammars and therefore uses all available categories to model the data. For the production distribution it creates two categories with equal posteriors to model the $S$ category, while for emissions the model collapses categories $A$ and $C$ into category $1$, and splits category $B$ over $3$ and $5$. This grammar is more expressive than the target grammar, over-generating but including the target grammar as a subset. The particular grammar found by the MLE model is dependent on the (random) initialisation and the fact that the EM algorithm can only find a local maximum, however it will always use all available categories to model the data.

**Chinese-English machine translation** The real-world translation experiment aims to determine whether the model can learn and generalise from a noisy large-scale parallel machine translation corpus, and provide performance benefits on the standard evaluation metrics. We evaluate our model on the IWSLT 2005 Chinese to English translation task [17], using the 2004 test set as development data for tuning the hyperparameters. The statistics for this data are presented in Table 2. The training data made available for this task consisted of 40k pairs of transcribed utterances, drawn from the travel domain. The translation phrase pairs that form the base of our grammar are induced using the standard alignment and translation phrase pair extraction heuristics used in phrase-based translation models [6]. As these heuristics aren't based on a generative model, and don't guarantee that the target translation will be reachable from the source, we discard those sentence pairs for which we cannot produce a derivation, leaving 33,164 sentences for training. Model performance is evaluated using the standard Bleu4 metric [18] which measures average $n$-gram precision, $n \leq 4$.

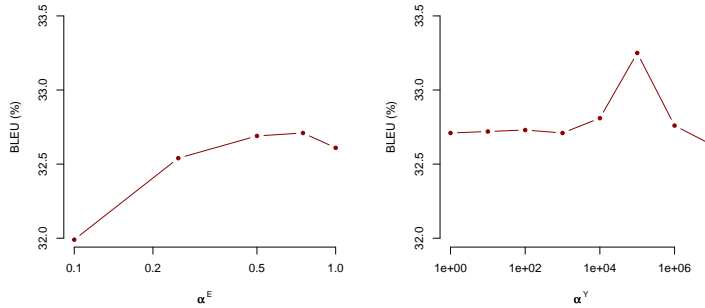

Figure 3: Tuning the Dirichlet $\alpha$ parameters for the emission and rule type distributions (development set).

| | MLE | Uniform $P_0$ | $P_0 = M_1$ | $P_0 = RF$ |
|---|---|---|---|---|
| Single Category | 32.9 | 35.5 | 37.1 | 38.7 |

Table 3: Test results for the model with a single non-terminal category and various emission priors ($B_{LEU}$).

| | MLE | $P_0 = RF$ |
|---|---|---|
| 5 Categories | 29.9 | 38.8 |

Table 4: Test set results for the hierarchical model with the variational distribution truncated at five non-terminal categories ($B_{LEU}$).

We first evaluate our model using a grammar with a single non-terminal category (rendering the hierarchical prior redundant) and vary the prior $P_0$ used for the emission parameters. For this model we investigate the effect that the emission and rule-type priors have on translation performance. Figure 3 graphs the variation in Bleu score versus the two free hyperparameters for the model with a simple uniform $P_0$, evaluated on the development corpus. Both graphs show a convex relationship, with $\alpha^Y$ being considerably more peaked. For the $\alpha^E$ hyperparameter the optimal value is 0.75, indicating that the emission distribution benefits from a slightly sparse distribution, but not far from the uniform value of 1.0. The sharp curve for the $\alpha^Y$ rule-type distribution hyperparameter confirms our earlier hypothesis that the model requires considerable smoothing in order to force it to place probability mass on long derivations rather than simply placing it all on the largest translation pairs.

The optimal hyperparameter values on the development data for the two structured emission distribution priors, Model 1 ($M^1$) and relative frequency ($RF$), also provide insight into the underlying models. The $M^1$ prior has a heavy bias towards smaller translation pairs, countering the model's inherent bias. Thus the optimal value for the $\alpha^Y$ parameter is 1.0, suggesting that the two biases balance. Conversely the $RF$ prior is biased towards larger translation pairs reinforcing the model's bias, thus a very large value ($10^6$) for the $\alpha^Y$ parameter gives optimal development set performance.

Table 3 shows the performance of the single category models with each of the priors on the test set.[4] The results show that all the Bayesian models outperform the MLE, and that non-uniform priors help considerably, with the $RF$ prior obtaining the highest score.

In Table 4 we show the results for taking the best performing $RF$ model from the previous experiment and increasing the variational approximation's truncation limit to five non-terminals. The $\alpha^P$ was set to 1.0, corresponding to a sparse distribution over binary productions.[5] Here we see that the HDP model improves slightly over the single category approximation. However the baseline MLE model uses the extra categories to overfit the training data significantly, resulting in much poorer generalisation performance.

# 5 Conclusion

We have proposed a Bayesian model for inducing synchronous grammars and demonstrated its efficacy on both synthetic and real machine translation tasks. The sophisticated priors over the model's parameters address limitations of MLE models, most notably overfitting, and effectively model the nature of the translation task. In addition, the incorporation of a hierarchical prior opens the door to the unsupervised induction of grammars capable of representing the latent structure of translation. Our Bayesian model of translation using synchronous grammars provides a basis upon which more sophisticated models can be built, enabling a move away from the current heuristically engineered translation systems.

## Footnotes

[1]Current translation systems more commonly use the conditional, rather than joint, estimator.

[2]The derivation specifies the translation. Alternatively we could bound on the likelihood of a translation, marginalising out the derivation. However, this bound cannot be maximised tractably when $\mathbf{e}$ is unobserved.

[3]No structured $P_0$ was used in this model, rather a simple Dirichlet prior with uniform $\alpha^E$ was employed for the emission distribution.

[4]For comparison, a state-of-the-art SCFG decoder based on the heuristic estimator, incorporating a trigram language model and using minimum error rate training achieves a $B_{LEU}$ score of approximately 46.

[5]As there are five non-terminal categories, an $\alpha^P = 5^2$ would correspond to a uniform distribution.

# References

[1] Andreas Zollmann and Ashish Venugopal. Syntax augmented machine translation via chart parsing. In *Proc. of the HLT-NAACL 2006 Workshop on Statistical Machine Translation*, New York City, June 2006.

[2] David Chiang. Hierarchical phrase-based translation. *Computational Linguistics*, 33(2):201–228, 2007.

[3] Percy Liang, Slav Petrov, Michael Jordan, and Dan Klein. The infinite PCFG using hierarchical Dirichlet processes. In *Proc. of the 2007 Conference on Empirical Methods in Natural Language Processing (EMNLP-2007)*, pages 688–697, Prague, Czech Republic, 2007.

[4] Jenny Rose Finkel, Trond Grenager, and Christopher D. Manning. The infinite tree. In *Proc. of the 45th Annual Meeting of the ACL (ACL-2007)*, Prague, Czech Republic, 2007.

[5] Y. W. Teh, M. I. Jordan, M. J. Beal, and D. M. Blei. Hierarchical Dirichlet processes. *Journal of the American Statistical Association*, 101(476):1566–1581, 2006.

[6] Philipp Koehn, Franz Josef Och, and Daniel Marcu. Statistical phrase-based translation. In *Proc. of the 3rd International Conference on Human Language Technology Research and 4th Annual Meeting of the NAACL (HLT-NAACL 2003)*, pages 81–88, Edmonton, Canada, May 2003.

[7] Michel Galley, Jonathan Graehl, Kevin Knight, Daniel Marcu, Steve DeNeefe, Wei Wang, and Ignacio Thayer. Scalable inference and training of context-rich syntactic translation models. In *Proc. of the 44th Annual Meeting of the ACL and 21st International Conference on Computational Linguistics (COLING/ACL-2006)*, pages 961–968, Sydney, Australia, July 2006.

[8] Dekai Wu. Stochastic inversion transduction grammars and bilingual parsing of parallel corpora. *Computational Linguistics*, 23(3):377–403, 1997.

[9] Colin Cherry and Dekany Lin. Inversion transduction grammar for joint phrasal translation modeling. In *Proc. of the HLT-NAACL Workshop on Syntax and Structure in Statistical Translation (SSST 2007)*, Rochester, USA, 2007.

[10] Hao Zhang, Chris Quirk, Robert C. Moore, and Daniel Gildea. Bayesian learning of non-compositional phrases with synchronous parsing. In *Proc. of the 46th Annual Conference of the Association for Computational Linguistics: Human Language Technologies (ACL-08:HLT)*, pages 97–105, Columbus, Ohio, June 2008.

[11] Daniel Marcu and William Wong. A phrase-based, joint probability model for statistical machine translation. In *Proc. of the 2002 Conference on Empirical Methods in Natural Language Processing (EMNLP-2002)*, pages 133–139, Philadelphia, July 2002. Association for Computational Linguistics.

[12] John DeNero, Dan Gillick, James Zhang, and Dan Klein. Why generative phrase models underperform surface heuristics. In *Proc. of the HLT-NAACL 2006 Workshop on Statistical Machine Translation*, pages 31–38, New York City, June 2006.

[13] Philip M. Lewis II and Richard E. Stearns. Syntax-directed transduction. *J. ACM*, 15(3):465–488, 1968.

[14] P. F. Brown, S. A. Della Pietra, V. J. Della Pietra, and R. L. Mercer. The mathematics of statistical machine translation: Parameter estimation. *Computational Linguistics*, 19(2):263–311, 1993.

[15] Mark Johnson. The DOP estimation method is biased and inconsistent. *Computational Linguistics*, 28(1):71–76, 2002.

[16] Matthew Beal. *Variational Algorithms for Approximate Bayesian Inference*. PhD thesis, The Gatsby Computational Neuroscience Unit, University College London, 2003.

[17] Matthias Eck and Chiori Hori. Overview of the IWSLT 2005 evaluation campaign. In *Proc. of the International Workshop on Spoken Language Translation*, Pittsburgh, October 2005.

[18] Kishore Papineni, Salim Roukos, Todd Ward, and Wei-Jing Zhu. Bleu: a method for automatic evaluation of machine translation. In *Proc. of the 40th Annual Meeting of the ACL and 3rd Annual Meeting of the NAACL (ACL-2002)*, pages 311–318, Philadelphia, Pennsylvania, 2002.
